# On Neural Networks with Minimal Weights

**Vasken Bohossian**          **Jehoshua Bruck**

California Institute of Technology
Mail Code 136-93
Pasadena, CA 91125
E-mail: {vincent, bruck}@paradise.caltech.edu

## Abstract

Linear threshold elements are the basic building blocks of artificial neural networks. A linear threshold element computes a function that is a sign of a weighted sum of the input variables. The weights are arbitrary integers; actually, they can be very big integers—exponential in the number of the input variables. However, in practice, it is difficult to implement big weights. In the present literature a distinction is made between the two extreme cases: linear threshold functions with polynomial-size weights as opposed to those with exponential-size weights. The main contribution of this paper is to fill up the gap by further refining that separation. Namely, we prove that the class of linear threshold functions with polynomial-size weights can be divided into subclasses according to the degree of the polynomial. In fact, we prove a more general result—that there exists a minimal weight linear threshold function for any arbitrary number of inputs and any weight size. To prove those results we have developed a novel technique for constructing linear threshold functions with minimal weights.

## 1  Introduction

Human brains are by far superior to computers for solving hard problems like combinatorial optimization and image and speech recognition, although their basic building blocks are several orders of magnitude slower. This observation has boosted interest in the field of artificial neural networks [Hopfield 82], [Rumelhart 82]. The latter are built by interconnecting multiple artificial neurons (or linear threshold gates), whose behavior is inspired by that of biological neurons. Artificial neural networks have found promising applications in pattern recognition, learning and

other data processing tasks. However most of the research has been oriented towards the practical aspect of neural networks, simulating or building networks for particular tasks and then comparing their performance with that of more traditional methods for those particular tasks. To compare neural networks to other computational models one needs to develop the theoretical settings in which to estimate their capabilities and limitations.

## 1.1   Linear Threshold Gate

The present paper focuses on the study of a single linear threshold gate (artificial neuron) with binary inputs and output as well as integer weights (synaptic coefficients). Such a gate is mathematically described by a *linear threshold function*.

**Definition 1** (Linear Threshold Function)
A linear threshold function of $n$ variables is a Boolean function $f : \{-1,1\}^n \to \{-1,1\}$ that can be written as

$$f(\vec{x}) = sgn(F(\vec{x})) = \begin{cases} 1 & \text{, for } F(\vec{x}) \geq 0 \\ -1 & \text{, otherwise} \end{cases} \quad \text{, where } F(\vec{x}) = \vec{w} \cdot \vec{x} = \sum_{i=1}^{n} w_i x_i$$

for any $\vec{x} \in \{-1,1\}^n$ and a fixed $\vec{w} \in Z^n$.

Although we could allow the weights $w_i$ to be real numbers, it is known [Muroga 71], [Raghavan 88] that for a, binary input neuron, one needs $O(n \log n)$ bits per weight, where $n$ is the number of inputs. So in the rest of the paper, we will assume without loss of generality that all weights are integers.

## 1.2   Motivation

Many experimental results in the area of neural networks have indicated that the magnitudes of the coefficients in the linear threshold elements grow very fast with the size of the inputs and therefore limit the practical use of the network. One natural question to ask is the following. How limited is the computational power of the network if one limits oneself to threshold elements with only "small" growth in the size of the coefficients? To answer that question we have to define a measure of the magnitudes of the weights. Note that, given a function $f$, the weight vector $\vec{w}$ is not unique (see Example 1 below).

**Definition 2** (Weight Space)
Given a linear threshold function $f$ we define $W$ as the set of all weights that satisfy Definition 1, that is $W = \{\vec{w} \in Z^n : \forall \vec{x} \in \{-1,1\}^n, sgn(\vec{w} \cdot \vec{x}) = f(\vec{x})\}$.

Here follows a measure of the size of the weights.

**Definition 3** (Minimal Weight Size)
We define the size of a weight vector as the sum of the absolute values of the weights. The minimal weight size of a linear threshold function is defined as :

$$S[f] = \min_{\vec{w} \in W} \left( \sum_{i=1}^{n} |w_i| \right)$$

The particular vector that achieves the minimum is called a minimal weight vector.

Naturally, $S[f]$ is a function of $n$.

It has been shown [Hastad 94], [Myhill 61], [Shawe-Taylor 92], [Siu 91] that there exists a linear threshold function that can be implemented by a single threshold element with exponentially growing weights, $S[f] \sim 2^n$, but cannot be implemented by a threshold element with smaller : polynomialy growing weights, $S[f] \sim n^d$, $d$ constant. In light of that result the above question was dealt with by defining a class within the set of linear threshold functions : the class of functions with "small" (i.e. polynomialy growing) weights [Siu 91]. Most of the recent research focuses on the power of circuits with small weights, relative to circuits with arbitrary weights [Goldmann 92], [Goldman 93]. Rather than dealing with circuits we are interested in studying a single threshold gate. The main contribution of the present paper is to further refine the division of small versus arbitrary weights. We separate the set of functions with small weights into classes indexed by $d$, the degree of polynomial growth and show that all of them are non-empty. In particular, we develop a technique for proving that a weight vector is minimal. We use that technique to construct a function of size $S[f] = s$ for an arbitrary $s$.

## 1.3 Approach

The main difficulty in analyzing the size of the weights of a threshold element is due to the fact that a single linear threshold function can be implemented by different sets of weights as shown in the following example.

**Example 1** (A Threshold Function with Minimal Weights)
Consider the following two sets of weights (weight vectors).

$$\vec{w}_1 = (1\ 2\ 4), \quad F_1(\vec{x}) = x_1 + 2x_2 + 4x_3$$

$$\vec{w}_2 = (2\ 4\ 8), \quad F_2(\vec{x}) = 2x_1 + 4x_2 + 8x_3$$

They both implement the same threshold function

$$f(\vec{x}) = sgn(F_2(\vec{x})) = sgn(2F_1(\vec{x})) = sgn(F_1(\vec{x}))$$

A closer look reveals that $f(\vec{x}) = sgn(x_3)$, implying that none of the above weight vectors has minimal size. Indeed, the minimal one is $\vec{w}_3 = (0\ 0\ 1)$ and $S[f] = 1$.

It is in general difficult to determine if a given set of weights is minimal [Amaldi 93], [Willis 63]. Our technique consists of limiting the study to only a particular subset of linear threshold functions, a subset for which it is possible to prove that a given weight vector is minimal. That subset is loosely defined by the requirement that there exist input vectors for which $f(\vec{x}) = f(-\vec{x})$. The existence of such a vector, called a *root* of $f$, puts a constraint on the weight vector used to implement $f$. The larger the set of roots – the larger the constraint on the set of weight vectors, which in turn helps determine the minimal one. A detailed description of the technique is given in Section 2.

## 1.4 Organization

Here follows a brief outline of the rest of the paper. Section 2 mathematically defines the setting of the problem as well as derives some basic results on the properties of functions that admit roots. Those results are used as building blocks for the proof of the main results in Section 3. It also introduces a construction method for functions with minimal weights. Section 3 presents the main result : for any weight size, $s$, and any number of inputs, $n$, there exists an $n$-input linear threshold function that requires weights of size $S[f] = s$. Section 4 presents some applications of the result of Section 3 and indicates future research directions.

## 2    Construction of Minimal Threshold Functions

The present section defines the mathematical tools used to construct functions with minimal weights.

### 2.1    Mathematical setting

We are interested in constructing functions for which the minimal weight is easily determined. Finding the minimal weight involves a search, we are therefore interested in finding functions with a constrained weight spaces. The following tools allows us to put constraints on $W$.

**Definition 4** (Root Space of a Boolean Function)
A vector $\vec{v} \in \{-1,1\}^n$ such that $f(\vec{v}) = f(-\vec{v})$ is called a root of $f$. We define the root space, $R$, as the set of all roots of $f$.

**Definition 5** (Root Generator Matrix)
For a given weight vector $\vec{w} \in W$ and a root $\vec{v} \in R$, the root generator matrix, $G = (g_{ij})$, is a $(n \times k)$-matrix, with entries in $\{-1, 0, 1\}$, whose rows $\vec{g}$ are orthogonal to $\vec{w}$ and equal to $\vec{v}$ at all non-zero coordinates, namely,

1. $G\vec{w} = \vec{0}$

2. $g_{ij} = 0$ or $g_{ij} = v_j$ for all $i$ and $j$.

**Example 2** (Root Generator Matrix)
Suppose that we are given a linear threshold function specified by a weight vector $\vec{w} = (1,1,2,4,1,1,2,4)$. By inspection we determine one root $\vec{v} = (1,1,1,1,-1,-1,-1,-1)$. Notice that $w_1 + w_2 - w_7 = 0$ which can be written as $\vec{g} \cdot \vec{w} = 0$, where $\vec{g} = (1,1,0,0,0,0,-1,0)$ is a row of $G$. Set $\vec{r} = \vec{v} - 2\vec{g}$. Since $\vec{g}$ is equal to $\vec{v}$ at all non-zero coordinates, $\vec{r} \in \{-1,1\}^n$. Also $\vec{r} \cdot \vec{w} = \vec{v} \cdot \vec{w} + \vec{g} \cdot \vec{w} = 0$. We have generated a new root : $\vec{r} = (-1,-1,1,1,-1,-1,1,-1)$.

**Lemma 6** (Orthogonality of $G$ and $W$)
For a given weight vector $\vec{w} \in W$ and a root $\vec{v} \in R$

$$\vec{u}G^T = \vec{0}$$

holds for any weight vector $\vec{u} \in W$.

**Proof.** For an arbitrary $\vec{u} \in W$ and an arbitrary row, $\vec{g}_i$, of $G$, let $\vec{v}' = \vec{v} - 2\vec{g}_i$. By definition of $\vec{g}_i$, $\vec{v}' \in \{-1,1\}^n$ and $\vec{v}' \cdot \vec{w} = 0$. That implies $f(\vec{v}') = f(-\vec{v}')$ : $\vec{v}'$ is a root of $f$. For any weight vector $\vec{u} \in W$, $sgn(\vec{u} \cdot \vec{v}') = sgn(-\vec{u} \cdot \vec{v}')$. Therefore $\vec{u} \cdot (\vec{v} - 2\vec{g}_i) = 0$ and finally, since $\vec{v} \cdot \vec{u} = 0$ we get $\vec{u} \cdot \vec{g}_i = 0$. $\square$

**Lemma 7** (Minimality)
For a given weight vector $\vec{w} \in W$ and a root $\vec{v} \in R$ if $rank(G) = n - 1$ (i.e. $G$ has $n - 1$ independent rows) and $|w_i| = 1$ for some $i$, then $\vec{w}$ is the minimal weight vector.

**Proof.** From Lemma 6 any weight vector $\vec{u}$ satisfies $\vec{u}G^T = \vec{0}$. $rank(G) = n - 1$ implies that $dim(W) = 1$, i.e. all possible weight vectors are integer multiples of each other. Since $|w_i| = 1$, all vectors are of the form $\vec{u} = k\vec{w}$, for $k \geq 1$. Therefore $\vec{w}$ has the smallest size. $\square$

We complete Example 2 with an application of Lemma 7.

**Example 3** (Minimality)
Given $\vec{w} = (1, 1, 2, 4, 1, 1, 2, 4)$ and $\vec{v} = (1, 1, 1, 1, -1, -1, -1, -1)$ we can construct :

$$G = \begin{pmatrix} 1 & 0 & 0 & 0 & -1 & 0 & 0 & 0 \\ 0 & 1 & 0 & 0 & 0 & -1 & 0 & 0 \\ 0 & 0 & 1 & 0 & 0 & 0 & -1 & 0 \\ 0 & 0 & 0 & 1 & 0 & 0 & 0 & -1 \\ 1 & 0 & 0 & 0 & 0 & -1 & 0 & 0 \\ 1 & 1 & 0 & 0 & 0 & 0 & -1 & 0 \\ 1 & 1 & 1 & 0 & 0 & 0 & 0 & -1 \end{pmatrix}$$

It is easy to verify that $rank(G) = n - 1 = 7$ and therefore, by Lemma 7, $\vec{w}$ is minimal and $S[f] = 16$.

## 2.2  Construction of minimal weight vectors

In Example 3 we saw how, given a weight vector, one can show that it is minimal. In this section we present an example of a linear threshold function with minimal weight size, with an arbitrary number of input variables.

We would like to construct a weight vector and show that it is minimal. Let the number of inputs, $n$, be even. Let $\vec{w}$ consist of two identical blocks : $(w_1, w_2, ..., w_{n/2}, w_1, w_2, ..., w_{n/2})$. Clearly, $\vec{v} = (1, 1, ..., 1, -1, -1, ..., -1)$ is a root and $G$ is the corresponding generator matrix.

$$G = \begin{pmatrix} 1 & 0 & 0 & 0 & ... & 0 & 0 & 0 & -1 & 0 & 0 & 0 & ... & 0 & 0 & 0 \\ 0 & 1 & 0 & 0 & ... & 0 & 0 & 0 & 0 & -1 & 0 & 0 & ... & 0 & 0 & 0 \\ 0 & 0 & 1 & 0 & ... & 0 & 0 & 0 & 0 & 0 & -1 & 0 & ... & 0 & 0 & 0 \\ \vdots & & & & & & & & & & & & & & & \vdots \\ 0 & 0 & 0 & 0 & ... & 0 & 1 & 0 & 0 & 0 & 0 & 0 & ... & 0 & -1 & 0 \\ 0 & 0 & 0 & 0 & ... & 0 & 0 & 1 & 0 & 0 & 0 & 0 & ... & 0 & 0 & -1 \end{pmatrix}$$

# 3  The Main Result

The following theorem states that given an integer $s$ and a number of variables $n$ there exists a function of $n$ variables and minimal weight size $s$.

**Theorem 8** (Main Result)
For any pair $(s, n)$ that satisfies

1. $n \leq s \leq \begin{cases} 2^{\frac{n}{2}} & \text{, for } n \text{ even} \\ 2^{\frac{n-1}{2}} + 2^{\frac{n-3}{2}} & \text{, for } n \text{ odd} \end{cases}$

2. $s$ even

there exists a linear threshold function of $n$ variables, $f$, with minimal weight size $S[f] = s$.

**Proof.** Given a pair $(s, n)$, that satisfies the above conditions we first construct a weight vector $\vec{w}$ that satisfies $\sum_{i=1}^{n} |w_i| = s$, then show that it is the minimal weight vector of the function $f(x) = sgn(\vec{w} \cdot \vec{x})$. The proof is shown only for $n$ even.

CONSTRUCTION.

1. Define $(a_1, a_2, ..., a_{n/2}) = (1, 1, ..., 1)$.

2. If $\sum_{i=1}^{n/2} a_i < s/2$ then increase by one the smallest $a_i$ such that $a_i < 2^{i-2}$. (In the case of a tie take the $w_i$ with smallest index $i$).

3. Repeat the previous step until $\sum_{i=1}^{n/2} a_i = s/2$ or $(a_1, a_2, ..., a_N) = (1, 1, 2, 4, ..., 2^{\frac{n}{2}-2})$.

4. Set $\vec{w} = (a_1, a_2, ..., a_{n/2}, a_1, a_2, ..., a_{n/2})$.

Because we increase the size by one unit at a time the algorithm will converge to the desired result for any integer $s$ that satisfies $n \leq s \leq 2^{\frac{n}{2}}$. We have a construction for any valid $(s, n)$ pair. Let us show that $\vec{w}$ is minimal.

MINIMALITY. Given that $\vec{w} = (a_1, a_2, ..., a_{n/2}, a_1, a_2, ..., a_{a/2})$ we find a root $\vec{v} = (1, 1, ..., 1, -1, -1, ..., -1)$ and $n/2$ rows of the generator matrix $G$ corresponding to the equations $w_i = w_{i+\frac{n}{2}}$. To form additional rows note that the first $k$ $a_i$'s are powers of two (where $k$ depends on $s$ and $n$). Those can be written as $a_i = \sum_{j=1}^{i-1} a_j$ and generate $k - 1$ rows. And finally note that all other $a_i$, $i > k$, are smaller than $2^{k+1}$. Hence, they can be written as a binary expansion $a_i = \sum_{j=1}^{k} \alpha_{ij} a_j$ where $\alpha_{ij} \in \{0, 1\}$. There are $\frac{n}{2} - k$ such weights. $G$ has a total of $n - 1$ independent rows. $rank(G) = n - 1$ and $w_1 = 1$, therefore by Lemma 7, $\vec{w}$ is minimal and $S[f] = s$. □

**Example 4** (A Function of 10 variables and size $S[f] = 26$)
We start with $\vec{a} = (1, 1, 1, 1, 1)$. We iterate : $(1, 1, 2, 1, 1)$, $(1, 1, 2, 2, 1)$, $(1, 1, 2, 2, 2)$, $(1, 1, 2, 3, 2)$, $(1, 1, 2, 3, 3)$, $(1, 1, 2, 4, 3)$, $(1, 1, 2, 4, 4)$, and finally $(1, 1, 2, 4, 5)$. The construction algorithm converges to $\vec{a} = (1, 1, 2, 4, 5)$. We claim that $\vec{w} = (\vec{a}, \vec{a}) = (1, 1, 2, 4, 5, 1, 1, 2, 4, 5)$ is minimal. Indeed, $\vec{v} = (1, 1, 1, 1, 1, -1, -1, -1, -1, -1)$ and

$$G = \begin{pmatrix}
1 & 0 & 0 & 0 & 0 & -1 & 0 & 0 & 0 & 0 \\
0 & 1 & 0 & 0 & 0 & 0 & -1 & 0 & 0 & 0 \\
0 & 0 & 1 & 0 & 0 & 0 & 0 & -1 & 0 & 0 \\
0 & 0 & 0 & 1 & 0 & 0 & 0 & 0 & -1 & 0 \\
0 & 0 & 0 & 0 & 1 & 0 & 0 & 0 & 0 & -1 \\
1 & 0 & 0 & 0 & 0 & 0 & -1 & 0 & 0 & 0 \\
1 & 1 & 0 & 0 & 0 & 0 & 0 & -1 & 0 & 0 \\
1 & 1 & 1 & 0 & 0 & 0 & 0 & 0 & -1 & 0 \\
1 & 0 & 0 & 1 & 0 & 0 & 0 & 0 & 0 & -1
\end{pmatrix}$$

is a matrix of rank 9.

**Example 5** (Functions with Polynomial Size)

This example shows an application of Theorem 8. We define $\widehat{LT}^{(d)}$ as the set of linear threshold functions for which $S[f] \leq n^d$. The Theorem states that for any even $n$ there exists a function $f$ of $n$ variables and minimum weight $S[f] = n^d$. The implication is that for all $d$, $\widehat{LT}^{(d-1)}$ is a proper subset of $\widehat{LT}^{(d)}$

# 4  Conclusions

We have shown that for any reasonable pair of integers $(n, s)$, where s is even, there exists a linear threshold function of $n$ variables with minimal weight size $S[f] = s$. We have developed a novel technique for constructing linear threshold functions with minimal weights that is based on the existence of root vectors. An interesting application of our method is the computation of a lower bound on the number of linear threshold functions [Smith 66]. In addition, our technique can help in studying the trade-offs between a number of important parameters associated with

linear threshold (neural) circuits, including, the number of elements, the number of layers, the fan-in, fan-out and the size of the weights.

## Acknowledgements

This work was supported in part by the NSF Young Investigator Award CCR-9457811, by the Sloan Research Fellowship, by a grant from the IBM Almaden Research Center, San Jose, California, by a grant from the AT&T Foundation and by the center for Neuromorphic Systems Engineering as a part of the National Science Foundation Engineering Research Center Program; and by the California Trade and Commerce Agency, Office of Strategic Technology.

## References

[Amaldi 93] E. Amaldi and V. Kann. The complexity and approximability of finding maximum feasible subsystems of linear relations. Ecole Polytechnique Federale De Lausanne Technical Report, *ORWP 93/11*, August 1993.

[Goldmann 92] M. Goldmann, J. Hastad, and A. Razborov. Majority gates vs. general weighted threshold gates. *Computational Complexity*, (2):277–300, 1992.

[Goldman 93] M. Goldmann and M. Karpinski. Simulating threshold circuits by majority circuits. In *Proc. 25th ACM STOC*, pages pp. 551–560, 1993.

[Hastad 94] J. Hastad. On the size of weights for threshold gates. *SIAM. J. Disc. Math.*, 7:484–492, 1994.

[Hopfield 82] J. Hopfield. Neural networks and physical systems with emergent collective computational abilities. *Proc. of the USA National Academy of Sciences*, 79:2554–2558, 1982.

[Muroga 71] M. Muroga. *Threshold Logic and its Applications*. Wiley-Interscience, 1971.

[Myhill 61] J. Myhill and W. H. Kautz. On the size of weights required for linear-input switching functions. *IRE Trans. Electronic Computers*, (EC10):pp. 288–290, 1961.

[Raghavan 88] P. Raghavan. Learning in threshold networks: a computational model and applications. Technical Report RC 13859, IBM Research, July 1988.

[Rumelhart 82] D. Rumelhart and J. McClelland. Parallel distributed processing: Explorations in the microstructure of cognition. *MIT Press*, 1982.

[Shawe-Taylor 92] J. S. Shawe-Taylor, M. H. G. Anthony, and W. Kern. Classes of feedforward neural networks and their circuit complexity. *Neural Networks*, Vol. 5:pp. 971–977, 1992.

[Siu 91] K. Siu and J. Bruck. On the power of threshold circuits with small weights. *SIAM J. Disc. Math.*, Vol. 4(No. 3):pp. 423–435, August 1991.

[Smith 66] D. R. Smith. Bounds on the number of threshold functions. *IEEE Transactions on Electronic Computers*, June 1966.

[Willis 63] D. G. Willis. Minimum weights for threshold switches. In *Switching Theory in Space Techniques*. Stanford University Press, Stanford, Calif., 1963.